# PAC-Bayes & Margins

**John Langford**
IBM Research
Email: jcl@cs.cmu.edu

**John Shawe-Taylor**
Royal Holloway, University of London
Email: jst@cs.rhul.ac.uk

## Abstract

We show two related things:

(1) Given a classifier which consists of a weighted sum of features with a large margin, we can construct a stochastic classifier with negligibly larger training error rate. The stochastic classifier has a future error rate bound that depends on the margin distribution and is independent of the size of the base hypothesis class.

(2) A new true error bound for classifiers with a margin which is simpler, functionally tighter, and more data-dependent than all previous bounds.

## 1 Introduction

PAC-Bayes bounds [8] (improved in [7] and again in [10]) are interesting for constructing bounds on future error rate in classification given only an assumption that examples are drawn independently from some (unknown) distribution. One drawback of PAC-Bayes bounds is that they only apply strongly to *stochastic* classifiers—classifiers which make randomized predictions. Most learning algorithms do not produce stochastic predictors, so the PAC-Bayes bound can only be applied to classifiers altered to form a stochastic classifier. This alteration can be done either by a sensitivity analysis [5] or by using the Bayesian posterior [10] directly.

Although stochastic classifiers are rare in practice, voting classifiers are quite common. These include "Bayes optimal" classifiers, Maximum Entropy classifiers [4], Adaboost [9], and others which has motivated analysis of true error bounds specialized to voting classifiers. In particular, we know that when the vote is decided by a margin of at least $\gamma$ on the training set, the true error rate is more tightly bound than might be naively expected. Unfortunately, these bounds are still unsatisfying for practical use because they often give the meaningless "future error rate is less than 1" predictions.

Here we analyze the connection between PAC-Bayes and margin bounds. In particular, we show that the "build a stochastic classifier" approach [5] to constructing a non-vacuous PAC-Bayes bound is always possible given that some margin $\gamma$ holds for the training set. This also shows that PAC-Bayes bounds are capable of capturing the low effective "sample complexity" which exists when a large margin classifier is used.

An alteration to the PAC-Bayes result proves a new bound for margins which is a

shorter argument and much tighter than previous margin bounds.

There are two mathematical flavors of margin bound dependent upon the weights $w_i$ of the vote and the features $x_i$ that the vote is taken over.

1. Those ([12], [1]) with a bound on $\sum_i w_i^2$ and $\sum_i x_i^2$ ("$l_2/l_2$" bounds).
2. Those ([11], [6]) with a bound on $\sum_i w_i$ and $\max_i x_i$ ("$l_1/l_\infty$" bounds).

The results here are of the "$l_2/l_2$" form. We improve on Shawe-Taylor et al. [12] and Bartlett [1] by a $\log(m)^2$ sample complexity factor and much tighter constants (1000 or unstated versus 9 or 18 as suggested by Section 2.2). In addition, the bound here covers margin errors without weakening the error-free case.

Herbrich and Graepel [3] moved significantly towards the approach adopted in our paper, but the methodology adopted meant that their result does not scale well to high dimensional feature spaces as the bound here (and earlier results) do.

The layout of our paper is simple - we first show how to construct a stochastic classifier with a good true error bound given a margin, and then construct a margin bound.

## 2 Margin Implies PAC-Bayes Bound

### 2.1 Notation and theorem

Consider a feature space $X$ which may be used to make predictions about the value in an output space $Y = \{-1, +1\}$. We use the notation $\mathbf{x} = (\mathbf{x}_1, \ldots, \mathbf{x}_N)$ to denote an $N$ dimensional vector. Let the vote of a voting classifier be given by:

$$v_{\mathbf{w}}(\mathbf{x}) = \mathbf{w}\mathbf{x} = \sum_i \mathbf{w}_i \mathbf{x}_i.$$

The classifier is given by $c(\mathbf{x}) = \text{sign}\,(v_{\mathbf{w}}(\mathbf{x}))$. The number of "margin violations" or "margin errors" at $\gamma$ is given by:

$$\hat{e}_\gamma(c) = \Pr_{(\mathbf{x},y)\sim U(S)} (y v_{\mathbf{w}}(\mathbf{x}) < \gamma),$$

where $U(S)$ is the uniform distribution over the sample set $S$.

For convenience, we assume $v_{\mathbf{x}}(\mathbf{x}) \leq 1$ and $v_{\mathbf{w}}(\mathbf{w}) \leq 1$. Without this assumption, our results scale as $\sqrt{v_{\mathbf{x}}(\mathbf{x})}\sqrt{v_{\mathbf{w}}(\mathbf{w})}/\gamma$ rather than $1/\gamma$.

Any margin bound applies to a vector $\mathbf{w}$ in $N$ dimensional space. For every example, we can decompose the example into a portion which is parallel to $\mathbf{w}$ and a portion which is perpendicular to $\mathbf{w}$.

$$\mathbf{x}_\top = \mathbf{x} - \frac{v_{\mathbf{w}}(\mathbf{x})}{\|\mathbf{w}\|^2}\mathbf{w} \quad \mathbf{x}_\| = \mathbf{x} - \mathbf{x}_\top$$

The argument is simple: we exhibit a "prior" over the weight space and a "posterior" over the weight space with an analytical form for the KL-divergence. The stochastic classifier defined by the posterior has a slightly larger empirical error and a small true error bound.

For the next theorem, let $\bar{F}(x) = 1 - \int_{-\infty}^{x} \frac{1}{\sqrt{2\pi}} e^{-x^2/2} dx$ be the tail probability of a Gaussian with mean 0 and variance 1. Also let

$$e_{Q(\mathbf{w},\gamma,\epsilon)} = \Pr_{(\mathbf{x},y)\sim D, h\sim Q(\mathbf{w},\gamma,\epsilon)} (h(\mathbf{x}) \neq y)$$

be the true error rate of a stochastic classifier with distribution $Q(\epsilon, \mathbf{w}, \gamma)$ dependent on a free parameter $\epsilon$, the weights $\mathbf{w}$ of an averaging classifier, and a margin $\gamma$.

**Theorem 2.1** *There exists a function $Q$ mapping a weight vector $\mathbf{w}$, margin $\gamma$, and value $\epsilon > 0$ to a distribution $Q(\mathbf{w}, \gamma, \epsilon)$ such that*

$$\Pr_{S \sim D^m} \left( \forall \mathbf{w}, \gamma, \epsilon: \ \mathrm{KL}(\hat{e}_\gamma(c) + \epsilon \| e_{Q(\mathbf{w}, \gamma, \epsilon)}) \leq \frac{\ln \frac{1}{\bar{F}\left(\frac{\bar{F}^{-1}(\epsilon)}{\gamma}\right)} + \ln \frac{m+1}{\delta}}{m} \right) \geq 1 - \delta$$

*where $\mathrm{KL}(q\|p) = q \ln \frac{q}{p} + (1-q) \ln \frac{1-q}{1-p}$ = the Kullback-Leibler divergence between two coins of bias $q < p$.*

## 2.2 Discussion

Theorem 2.1 shows that when a margin exists it is *always* possible to find a "posterior" distribution (in the style of [5]) which introduces only a small amount of additional training error rate. The true error bound for this stochastization of the large-margin classifier is not dependent on the dimensionality except via the margin.

Since the Gaussian tail decreases exponentially, the value of $\bar{F}^{-1}(\epsilon)$ is not very large for any reasonable value of $\epsilon$. In particular, at $\bar{F}(3)$, we have $\epsilon \leq 0.01$. Thus, for the purpose of understanding, we can replace $\bar{F}^{-1}(\epsilon)$ with 3 and consider $\epsilon \simeq 0$. One useful approximation for $\bar{F}(x)$ with large $x$ is:

$$\bar{F}(x) \simeq \frac{e^{-x^2/2}}{\sqrt{2\pi}} (1/x)$$

If there are no margin errors $\hat{e}_\gamma(c) = 0$, then these approximations, yield the approximate bound:

$$\Pr_{S \sim D^m} \left( e_{Q(\mathbf{w}, \gamma, 0)} \leq \frac{\frac{9}{2\gamma^2} + \ln \frac{3\sqrt{2\pi}}{\gamma} + \ln \frac{m+1}{\delta}}{m} \right) \geq 1 - \delta$$

In particular, for large $m$ the true error is approximately bounded by $\frac{9}{2\gamma^2 m}$.

As an example, if $\gamma = 0.25$, the bound is less than 1 around $m = 100$ examples and less than 0.5 around $m = 200$ examples.

Later we show (see Lemmas 4.1 and 4.2 or Theorem 4.3) that the generalisation error of the original averaging classifier is only a factor 2 or 4 larger than that of the stochastic classifiers considered here. Hence, the bounds of Theorems 2.1 and 3.1 also give bounds on the averaging classifiers $\mathbf{w}$.

This theorem is robust in the presence of noise and margin errors. Since the PAC-Bayes bound works for *any* "posterior" $Q$, we are free to choose $Q$ dependent upon the data in any way. In practice, it may be desirable to follow an approach similar to [5] and allow the data to determine the "right" posterior $Q$. Using the data rather than the margin $\gamma$ allows the bound to take into account a fortuitous data distribution and robust behavior in the presence of a "soft margin" (a margin with errors). This is developed (along with a full proof) in the next section.

## 3  Main Full Result

We now present the main result. Here we state a bound which can take into account the distribution of the training set. Theorem 2.1 is a simple consequence

of this result. This theorem demonstrates the flexibility of the technique since it incorporates significantly more data-dependent information into the bound calculation. When applying the bound one would choose $\mu$ to make the inequality (1) an equality. Hence, any choice of $\mu$ determines $\epsilon$ and hence the overall bound. We then have the freedom to choose $\mu$ to optimise the bound.

As noted earlier, given a weight vector $\mathbf{w}$, any particular feature vector $\mathbf{x}$ decomposes into a portion $\mathbf{x}_\parallel$ which is parallel to $\mathbf{w}$ and a portion $\mathbf{x}_\top$ which is perpendicular to $\mathbf{w}$. Hence, we can write $\mathbf{x} = x_\parallel \mathbf{e}_\parallel + x_\top \mathbf{e}_\top$, where $\mathbf{e}_\parallel$ is a unit vector in the direction of $\mathbf{w}$ and $\mathbf{e}_\top$ is a unit vector in the direction of $\mathbf{x}_\top$. Note that we may have $yx_\parallel < 0$, if $\mathbf{x}$ is misclassified by $\mathbf{w}$.

**Theorem 3.1** *For all averaging classifiers $c$ with normalized weights $\mathbf{w}$ and for all $\epsilon > 0$ stochastic error rates, If we choose $\mu > 0$ such that*

$$E_{\mathbf{x},y \sim S} \bar{F} \left( \frac{yx_\parallel}{x_\top} \mu \right) = \epsilon \tag{1}$$

*then there exists a posterior distribution $Q(\mathbf{w}, \mu, \epsilon)$ such that*

$$\Pr_{S \sim D^m} \left( \forall \epsilon, \mathbf{w}, \mu : \ \mathrm{KL}(\epsilon \| e_{Q(\mathbf{w},\mu,\epsilon)}) \leq \frac{\ln \frac{1}{\bar{F}(\mu)} + \ln \frac{m+1}{\delta}}{m} \right) \geq 1 - \delta$$

*where $\mathrm{KL}(q\|p) = q \ln \frac{q}{p} + (1-q) \ln \frac{1-q}{1-p} =$ the Kullback-Leibler divergence between two coins of bias $q < p$.*

**Proof.** The proof uses the PAC-Bayes bound, which states that for all prior distributions $P$,

$$\Pr_{S \sim D^m} \left( \forall Q : \ \mathrm{KL}(\hat{e}_Q \| e_Q) \leq \frac{\mathrm{KL}(Q\|P) + \ln \frac{m+1}{\delta}}{m} \right) \geq 1 - \delta$$

We choose $P = N(0, I)$, an isotropic Gaussian[1].

A choice of the "posterior" $Q$ completes the proof. The $Q$ we choose depends upon the direction $\mathbf{w}$, the margin $\gamma$, and the stochastic error $\epsilon$. In particular, $Q$ equals $P$ in every direction perpendicular to $\mathbf{w}$, and a rectified Gaussian tail in the $\mathbf{w}$ direction[2]. The distribution of a rectified Gaussian tail is given by $R(\mu) = 0$ for $x < \mu$ and $R(\mu) = \frac{1}{\bar{F}(\mu)\sqrt{2\pi}} e^{-x^2/2}$ for $x \geq \mu$.

The chain rule for relative entropy (Theorem 2.5.3 of [2]) and the independence of draws in each dimension implies that:

$$
\begin{aligned}
\mathrm{KL}(Q\|P) &= \mathrm{KL}(Q_\parallel \| P_\parallel) + \mathrm{KL}(Q_\top \| P_\top) \\
&= \mathrm{KL}(R(\mu)\|N(0,1)) + \mathrm{KL}(P_\top \| P_\top) \\
&= \mathrm{KL}(R(\mu)\|N(0,1)) + 0 \\
&= \int_\mu^\infty \ln \frac{1}{\bar{F}(\mu)} R(x) dx \\
&= \ln \frac{1}{\bar{F}(\mu)}
\end{aligned}
$$

Thus, our choice of posterior implies the theorem if the empirical error rate is $\hat{e}_{Q(\mathbf{w},\mathbf{x},\epsilon)} \leq E_{\mathbf{x},y \sim S}\bar{F}\left(\frac{\mathbf{x}_\parallel}{\mathbf{x}_\top}\mu\right) \leq \epsilon$ which we show next.

Given a point $\mathbf{x}$, our choice of posterior implies that we can decompose the stochastic weight vector, $\hat{\mathbf{w}} = \hat{w}_\parallel \mathbf{e}_\parallel + \hat{w}_\top \mathbf{e}_\top + \tilde{\mathbf{w}}$, where $\mathbf{e}_\parallel$ is parallel to $\mathbf{w}$, $\mathbf{e}_\top$ is parallel to $\mathbf{x}_\top$ and $\tilde{\mathbf{w}}$ is a residual vector perpendicular to both. By our definition of the stochastic generation $\hat{\mathbf{w}}_\parallel \sim R(\mu)$ and $\hat{\mathbf{w}}_\top \sim N(0,1)$. To avoid an error, we must have:

$$
\begin{aligned}
y &= \text{sign}(v_{\hat{\mathbf{w}}}(\mathbf{x})) \\
&= \text{sign}(\hat{w}_\parallel x_\parallel + \hat{w}_\top x_\top).
\end{aligned}
$$

Then, since $\hat{w}_\parallel \geq \mu$, no error occurs if:

$$
y(\mu x_\parallel + \hat{w}_\top x_\top) > 0
$$

Since $\hat{w}_\top$ is drawn from $N(0,1)$ the probability of this event is:

$$
\Pr\left(y(\mu x_\parallel + \hat{w}_\top x_\top) > 0\right) \geq 1 - \bar{F}\left(\frac{y x_\parallel}{x_\top}\mu\right)
$$

And so, the empirical error rate of the stochastic classifier is bounded by:

$$
\hat{e}_Q \leq E_{\mathbf{x},y \sim S}\bar{F}\left(\frac{y x_\parallel}{x_\top}\mu\right) = \epsilon
$$

as required. ∎

### 3.1 Proof of Theorem 2.1

**Proof.** (sketch) The theorem follows from a relaxation of Theorem 3.1. In particular, we treat every example with a margin less than $\gamma$ as an error and use the bounds $\|\mathbf{x}_\top\| \leq 1$ and $\|\mathbf{x}_\parallel\| \geq \gamma$. ∎

### 3.2 Further results

Several aspects of the Theorem 3.1 appear arbitrary, but they are not. In particular, the choice of "prior" is not that arbitrary as the following lemma indicates.

**Lemma 3.2** *The set of $P$ satisfying $\exists P_{\|\|} : P(\mathbf{x}) = P_{\|\|}(\|\mathbf{x}\|^2)$ (rotational invariance) and $P(\mathbf{x}) = \prod_{i=1}^N p_i(\mathbf{x}_i)$ (independence of each dimension) is $N(0,\lambda I)$ for $\lambda > 0$.*

**Proof.** Rotational invariance together with the dimension independence imply that for all $i,j,x : p_i(x) = p_j(x)$ which implies that:

$$
P(\mathbf{x}) = \prod_{i=1}^N p(\mathbf{x}_i)
$$

for some function $p(\cdot)$. Applying rotational invariance, we have that:

$$
P(\mathbf{x}) = P_{\|\|}(\|\mathbf{x}\|^2) = \prod_{i=1}^N p(\mathbf{x}_i)
$$

This implies:

$$
\log P_{\|\|}\left(\sum_{i=1}^N \mathbf{x}_i^2\right) = \sum_{i=1}^N \log p(\mathbf{x}_i).
$$

Taking the derivative of this equation with respect to $\mathbf{x}_i$ gives

$$\frac{P'_{\||\|}(\|\mathbf{x}\|^2)2\mathbf{x}_i}{P_{\||\|}(\|\mathbf{x}\|^2)} = \frac{p'(\mathbf{x}_i)}{p(\mathbf{x}_i)}.$$

Since this holds for all values of $\mathbf{x}$ we must have

$$P_{\||\|}(t) = \lambda P'_{\||\|}(t)$$

for some constant $\lambda$, or $P_{\||\|}(t) = C\exp(\lambda t)$, for some constant $C$. Hence, $P(\mathbf{x}) = C\exp(\lambda\|\mathbf{x}\|^2)$, as required. ∎

The constant $\lambda$ in the previous lemma is a free parameter. However, the results do not depend upon the precise value of $\lambda$ so we choose 1 for simplicity. Some freedom in the choice of the "posterior" $Q$ does exist and the results are dependent on this choice. A rectified gaussian appears simplest.

## 4  Margin Implies Margin Bound

There are two methods for constructing a margin bound for the original averaging classifier. The first method is simplest while the second is sometimes significantly tighter.

### 4.1  Simple Margin Bound

First we note a trivial bound arising from a folk theorem and the relationship to our result.

**Lemma 4.1** *(Simple Averaging bound) For any stochastic classifier with distribution $Q$ and true error rate $e_Q$, the averaging classifier,*

$$c_Q(\mathbf{x}) = sign\left(\int_H h(\mathbf{x})dQ(h)\right)$$

*has true error rate:*

$$e(c_Q) \le 2e_Q$$

**Proof.** For every example $(\mathbf{x}, y)$, every time the averaging classifier errs, the probability of the stochastic classifier erring must be at least $1/2$. ∎

This result is interesting and of practical use when the empirical error rate of the original averaging classifier is low. Furthermore, we can prove that $c_Q(\mathbf{x})$ *is* the original averaging classifier.

**Lemma 4.2** *For $Q = Q(\mathbf{w}, \gamma, \epsilon)$ derived according to Theorems 2.1 and 3.1 and $c_Q(\mathbf{x})$ as in lemma 4.1:*
$$c_Q(\mathbf{x}) = sign\left(v_{\mathbf{w}}(\mathbf{x})\right)$$

**Proof.** For every $\mathbf{x}$ this equation holds because of two simple facts:

1. For any $\hat{\mathbf{w}}$ that classifies an input $\mathbf{x}$ differently from the averaging classifier, there is a unique equiprobable paired weight vector that agrees with the averaging classifier.

2. If $v_{\mathbf{w}}(\mathbf{x}) \ne 0$, then there exists a nonzero measure of classifier pairs which always agrees with the averaging classifier.

Condition (1) is met by reversing the sign of $\hat{\mathbf{w}}_T$ and noting that either the original random vector or the reversed random vector must agree with the averaging classifier.

Condition (2) is met by the randomly drawn classifier $\hat{\mathbf{w}} = \lambda \mathbf{w}$ and nearby classifiers for any $\lambda > 0$. Since the example is not on the hyperplane, there exists some small sphere of paired classifiers (in the sense of condition (1)). This sphere has a positive measure. ∎

The simple averaging bound is elegant, but it breaks down when the empirical error is large because:

$$e(c) \leq 2e_Q = 2(\hat{e}_Q + \Delta_m) \simeq 2\hat{e}_\gamma(c) + 2\Delta_m$$

where $\hat{e}_Q$ is the empirical error rate of a stochastic classifier and $\Delta_m$ goes to zero as $m \to \infty$. Next, we construct a bound of the form $e(c_Q) \leq \hat{e}_\gamma(c) + \Delta'_m$ where $\Delta'_m > \Delta_m$ but $\hat{e}_\gamma(c) \leq 2\hat{e}_\gamma(c)$.

## 4.2 A (Sometimes) Tighter Bound

By altering our choice of $\mu$ and our notion of "error" we can construct a bound which holds *without* randomization. In particular, we have the following theorem:

**Theorem 4.3** *For all averaging classifiers c with normalized weights $\mathbf{w}$ for all $\epsilon > 0$ "extra" error rates and $\gamma > 0$ margins:*

$$\Pr_{S \sim D^m}\left(\forall \epsilon, \mathbf{w}, \gamma: \ \mathrm{KL}(\hat{e}_\gamma(c) + \epsilon \| e(c) - \epsilon) \leq \frac{\ln \frac{1}{\bar{F}\left(\frac{2\bar{F}^{-1}(\epsilon)}{\gamma}\right)} + 2\ln \frac{m+1}{\delta}}{m}\right) \geq 1 - \delta$$

*where $\mathrm{KL}(q\|p) = q \ln \frac{q}{p} + (1-q)\ln \frac{1-q}{1-p}$ = the Kullback-Leibler divergence between two coins of bias $q < p$.*

The proof of this statement is strongly related to the proof given in [11] but noticeably simpler. It is also very related to the proof of theorem 2.1.

**Proof.** (sketch) Instead of choosing $\hat{\mathbf{w}}_\|$ so that the empirical error rate is increased by $\epsilon$, we instead choose $\hat{\mathbf{w}}_\|$ so that the number of margin violations at margin $\frac{\gamma}{2}$ is increased by at most $\epsilon$. This can be done by drawing from a distribution such as

$$\hat{\mathbf{w}}_\| \sim R\left(\frac{2\bar{F}^{-1}(\epsilon)}{\gamma}\right)$$

Applying the PAC-Bayes bound to this we reach a bound on the number of margin violations at $\frac{\gamma}{2}$ for the true distribution. In particular, we have:

$$\Pr_{S \sim D^m}\left(\mathrm{KL}\left(\hat{e}_\gamma(c) + \epsilon \| e_{Q,\frac{\gamma}{2}}\right) \leq \frac{\ln \frac{1}{\bar{F}\left(\frac{2\bar{F}^{-1}(\epsilon)}{\gamma}\right)} + \ln \frac{m+1}{\delta}}{m}\right) \geq 1 - \delta$$

The application is tricky because the bound does not hold uniformly for all $\gamma$.[3] Instead we can discretize $\gamma$ at scale $1/m$ and apply a union bound to get $\delta \to \delta/m+1$.

For any fixed example, $(\mathbf{x}, y)$ with probability $1 - \delta$, we know that with probability at least $1 - e_{Q,\frac{\gamma}{2}}$, the example has a margin of at least $\frac{\gamma}{2}$. Since the example has

a margin of at least $\frac{\gamma}{2}$ and our randomization doesn't change the margin by more than $\frac{\gamma}{2}$ with probability $1 - \epsilon$, the averaging classifier almost always predicts in the same way as the stochastic classifier implying the theorem. ■

## 4.3 Discussion & Open Problems

The bound we have obtained here is considerably tighter than previous bounds for averaging classifiers—in fact it is tight enough to consider applying to real learning problems and using the results in decision making.

Can this argument be improved? The simple averaging bound (lemma 4.1) and the margin bound (theorem 4.3) each have a regime in which they dominate. We expect that there exists some natural theorem which does well in both regimes simultaneously.

In order to verify that the margin bound is as tight as possible, it would also be instructive to study lower bounds.

## 4.4 Acknowledgements

Many thanks to David McAllester for critical reading and comments.

## Footnotes

[1]Later, the fact that an isotropic Gaussian has the same representation in all rotations of the coordinate sytem will be useful.

[2]Note that we use the invariance under rotation of $N(0, I)$ here to line up one dimension with $\mathbf{w}$.

[3]Thanks to David McAllester for pointing this out.

# References

[1] P. L. Bartlett, "The sample complexity of pattern classification with neural networks: the size of the weights is more important than the size of the network," *IEEE Transactions on Information Theory*, vol. 44, no. 2, pp. 525–536, 1998.

[2] Thomas Cover and Joy Thomas, "Elements of Information Theory" Wiley, New York 1991.

[3] Ralf Herbrich and Thore Graepel, A PAC-Bayesian Margin Bound for Linear Classifiers: Why SVMs work. In Advances in Neural Information Processing Systems 13, pages 224-230. 2001.

[4] T. Jaakkola, M. Meila, T. Jebara, "Maximum Entropy D iscrimination\char" NIPS 1999.

[5] John Langford and Rich Caruana, (Not) Bounding the True Error NIPS2001.

[6] John Langford, Matthias Seeger, and Nimrod Megiddo, "An Improved Predictive Accuracy Bound for Averaging Classifiers" ICML2001.

[7] John Langford and Matthias Seeger, "Bounds for Averaging Classifiers." CMU tech report, CMU-CS-01-102, 2001.

[8] David McAllester, "PAC-Bayesian Model Averaging" COLT 1999.

[9] Yoav Freund and Robert E. Schapire, "A Decision Theoretic Generalization of On-line Learning and an Application to Boosting" Eurocolt 1995.

[10] Matthias Seeger, "PAC-Bayesian Generalization Error Bounds for Gaussian Processes", Tech Report, Division of Informatics report EDI-INF-RR-0094. http://www.dai.ed.ac.uk/homes/seeger/papers/gpmcall-tr.ps.gz

[11] Robert E. Schapire, Yoav Freund, Peter Bartlett, and Wee Sun Lee, "Boosting the Margin: A new explanation for the effectiveness of voting methods" The Annals of Statistics, 26(5):1651-1686, 1998.

[12] J. Shawe-Taylor, P. L. Bartlett, R. C. Williamson, and M. Anthony. Structural risk minimization over data-dependent hierarchies. *IEEE Transactions on Information Theory*, 44(5):1926–1940, 1998.
